# Optimizing Classifiers for Imbalanced Training Sets

**Grigoris Karakoulas**
Global Analytics Group
Canadian Imperial Bank of Commerce
161 Bay St., BCE-11,
Toronto ON, Canada M5J 2S8
Email: karakoul@cibc.ca

**John Shawe-Taylor**
Department of Computer Science
Royal Holloway, University of London
Egham, TW20 0EX
England
Email: jst@dcs.rhbnc.ac.uk

## Abstract

Following recent results [9, 8] showing the importance of the fat-shattering dimension in explaining the beneficial effect of a large margin on generalization performance, the current paper investigates the implications of these results for the case of imbalanced datasets and develops two approaches to setting the threshold. The approaches are incorporated into ThetaBoost, a boosting algorithm for dealing with unequal loss functions. The performance of ThetaBoost and the two approaches are tested experimentally.

**Keywords:** Computational Learning Theory, Generalization, fat-shattering, large margin, pac estimates, unequal loss, imbalanced datasets

## 1 Introduction

Shawe-Taylor [8] demonstrated that the output margin can also be used as an estimate of the confidence with which a particular classification is made. In other words if a new example has an output value well clear of the threshold we can be more confident of the associated classification than when the output value is closer to the threshold. The current paper applies this result to the case where there are different losses associated with a false positive, than with a false negative. If a significant number of data points are misclassified we can use the criterion of minimising the empirical loss. If, however, the data is correctly classified the empirical loss is zero for all correctly separating hyperplanes. It is in this case that the approach can provide insight into how to choose the hyperplane and threshold. In summary, the paper suggests ways in which a hyperplane should be optimised for imbalanced datasets where the loss associated with misclassifying the less prevalent class is higher.

## 2   Background to the Analysis

**Definition 2.1** *[3] Let $\mathcal{F}$ be a set of real-valued functions. We say that a set of points $X$ is $\gamma$-shattered by $\mathcal{F}$ if there are real numbers $r_x$ indexed by $x \in X$ such that for all binary vectors $b$ indexed by $X$, there is a function $f_b \in \mathcal{F}$ realising dichotomy $b$ with margin $\gamma$. The fat-shattering dimension $\mathrm{Fat}_{\mathcal{F}}$ of the set $\mathcal{F}$ is a function from the positive real numbers to the integers which maps a value $\gamma$ to the size of the largest $\gamma$-shattered set, if this is finite, or infinity otherwise.*

In general we are concerned with classifications obtained by thresholding real-valued functions. The classification values will be $\{-1, 1\}$ instead of the usual $\{0, 1\}$ in order to simplify some expressions. Hence, typically we will consider a set $\mathcal{F}$ of functions mapping from an input space $X$ to the reals. For the sake of simplifying the presentation of our results we will assume that the threshold used for classification is 0. The results can be extended to other thresholds without difficulty. Hence we implicitly use the classification functions $H = T(\mathcal{F}) = \{T(f) : f \in \mathcal{F}\}$, where $T(f)$ is the function $f$ thresholded at 0. We will say that $f$ has $\gamma$ margin on the training set $\{(x_i, y_i) : i = 1, \ldots, m\}$, if $\min_{1 \le i \le m}\{y_i f(x_i)\} = \gamma$. Note that a positive margin implies that $T(f)$ is consistent.

**Definition 2.2** *Given a real-valued function $f : X \to [-1, 1]$ used for classification by thresholding at 0, and probability distribution $P$ on $X \times \{-1, 1\}$, we use $\mathrm{er}_P(f)$ to denote the following probability $\mathrm{er}_P(f) = P\{(x, y) : yf(x) \le 0\}$. Further suppose $0 \le \eta \le 1$, then we use $\mathrm{er}_P(f|\eta)$ to denote the probability*

$$\mathrm{er}_P(f|\eta) = P\{(x, y) : yf(x) \le 0 \| f(x)| \ge \eta\}.$$

The probability $\mathrm{er}_P(f|\eta)$ is the probability of misclassification of a randomly chosen example given that it has a margin of $\eta$ or more.

We consider the following restriction on the set of real-valued functions.

**Definition 2.3** *The real-valued function class $\mathcal{F}$ is closed under addition of constants if*

$$\eta \in \mathbb{R}, f \in \mathcal{F} \Rightarrow f + \eta \in \mathcal{F}.$$

Note that the linear functions (with threshold weights) used in perceptrons [9] satisfy this property as do neural networks with linear output units. Hence, this property applies to the Support Vector Machine, and the neural network examples. We now quote a result from [8].

**Theorem 2.4** *[8] Let $\mathcal{F}$ be a class of real-valued functions closed under addition of constants with fat-shattering dimension bounded by $\mathrm{Fat}_{\mathcal{F}}(\gamma)$ which is continuous from the right. With probability at least $1 - \delta$ over the choice of a random $m$ sample $(x_i, y_i)$ drawn according to $P$ the following holds. Suppose that for some $f \in \mathcal{F}$, $\eta > 0$,*

   *1. $y_i f(x_i) \ge -\eta + 2\gamma$ for all $(x_i, y_i)$ in the sample,*

   *2. $n = |\{i : y_i f(x_i) \ge \eta + 2\gamma\}|$,*

   *3. $n \ge 3\sqrt{2m(2d\ln(288m)\log_2(12em) + \ln(32m^2/\delta))}$,*

*Let $d = \mathrm{Fat}_{\mathcal{F}}(\gamma/6)$. Then the probability that a new example with margin $\eta$ is misclassified is bounded by*

$$\frac{3}{n}\left(2d\log_2(288m)\log_2(12em) + \log_2\frac{32m^2}{\delta}\right).$$

## 3  Unequal Loss Functions

We consider the situation where the loss associated with an example is different for misclassification of positive and negative examples. Let $L_h(x, y)$ be the loss associated with the classification function $h$ on example $(x, y)$. For the analysis considered above the loss function is taken to be $L_h(x, y) = |h(x) - y|$, that is 1 if the point $x$ is misclassified and 0 otherwise. This is also known as the discrete loss. In this paper we consider a different loss function for classification functions.

**Definition 3.1** *The loss function $L^\beta$ is defined as $L^\beta(x, y) = \beta y + (1 - y)$, if $h(x) \neq y$, and 0, otherwise.*

We first consider the classical approach of minimizing the empirical loss, that is the loss on the training set. Since, the loss function is no longer binary the standard theoretical results that can be applied are much weaker than for the binary case. The algorithmic implications will, however, be investigated under the assumption we are using a hyperplane parallel to the maximal margin hyperplane. The empirical risk is given by $ER(h) = \sum_{i=1}^{m} L^\beta(x_i, y_i)$, for the training set $\{(x_i, y_i) : i = 1, \ldots, m\}$.

Assuming that the training set can be correctly classified by the hypothesis class this criterion will not be able to distinguish between consistent hypotheses, hence giving no reason not to choose the standard maximal margin choice. However, there is a natural way to introduce the different losses into the maximal margin quadratic programming procedure [1]. Here, the constraints given are specified as $y_i(\langle w \cdot x_i \rangle + \theta) \geq 1, i = 1, 2, \ldots, m$. In order to force the hyperplane away from the positive points which will incur greater loss, a natural heuristic is to set $y_i = -1$ for negative examples and $y_i = 1/\beta$ for positive points, hence making them further from the decision boundary. In the case where consistent classification is possible, the effect of this will be to move the hyperplane parallel to itself so that the margin on the positive side is $\beta$ times that on the negative side. Hence, to solve the problem we simply use the standard maximal margin algorithm [1] and then replace the threshold $\theta$ with

$$b = \frac{1}{1 + \beta}[(w \cdot x^+) + \beta(w \cdot x^-)], \tag{1}$$

where $x^+$ ($x^-$) is one of the closest positive (negative) points.

The alternative approach we wish to employ is to consider other movements of the hyperplane parallel to itself while retaining consistency. Let $\gamma_0$ be the margin of the maximal margin hyperplane. We consider a consistent hyperplane $h_\eta$ with margin $\gamma_0 + \eta$ to the positive examples, and $\gamma_0 - \eta$ to the negative example. The basic analytic tool is Theorem 2.4 which will be applied once for the positive examples and once for the negative examples (note that classifications are in the set $\{-1, 1\}$).

**Theorem 3.2** *Let $h_0$ be the maximal margin hyperplane with margin $\gamma_0$, while $h_\eta$ is as above with $\eta < \gamma_0$. Set $\gamma^+ = (\gamma_0 + \eta)/2$ and $\gamma^- = (\gamma_0 - \eta)/2$. With probability at least $1 - \delta$ over the choice of a random $m$ sample $(x_i, y_i)$ drawn according to $P$ the following holds. Suppose that for $h_0$*

  *1. $n_0 = |\{i : y_i h_0(x_i) \geq 2\eta + \gamma_0\}|$,*

  *2. $n_0 \geq 3\sqrt{2m(d \ln(288m) \log_2(12em) + \ln(8/\delta))}$,*

*Let $d^+ = \text{Fat}_{\mathcal{F}}(\gamma^+/6)$ and $d^- = \text{Fat}_{\mathcal{F}}(\gamma^-/6)$. Then we can bound the expected loss by*

$$\frac{3}{n_0}(2 \max(\beta d^+, d^-) \log_2(288m) \log_2(12em) + \beta \log_2(32m^2/\delta))$$

**Proof**: Using Theorem 2.4 we can bound the probability of error given that the correct classification is positive in terms of the expression with the fat shattering dimension $d^+$ and $n = n_0$, while for a negative example we can bound the probability of error in terms of the expression with fat shattering dimension $d^-$ and $n = m$. Hence, the expected loss can be bounded by taking the maximum of the second bound with $n^+$ in place of $m$ together with a factor $\beta$ in front of the second log term and the first bound multiplied by $\beta$. ∎

The bound obtained suggests a way of optimising the choice of $\eta$, namely to minimise the expression for the fat shattering dimension of linear functions [9]. Solving for $\eta$ in terms of $\gamma_0$ and $\beta$ gives

$$\eta = \gamma_0 \left( (\sqrt[3]{\beta} - 1)/(\sqrt[3]{\beta} + 1) \right). \tag{2}$$

This choice of $\eta$ does not in general agree with that suggested by the choice of the threshold $b$ in the previous section. In a later section we report on initial experiments for investigating the performance of these different choices.

## 4 The ThetaBoost Algorithm

The above idea for adjusting the margin in the case of unequal loss function can also be applied to the AdaBoost algorithm [2] which has been shown to maximise the margin on the training examples and hence the generalization can be bounded in terms of the margin and the fat-shattering dimension of the functions that can be produced by the algorithm [6]. We will first develop a boosting algorithm for unequal loss functions and then extend it for adjustable margin. More specifically, assume: (i) a set of training examples $(x_1, y_1), ..., (x_m, y_m)$ where $x_i \in X$ and $y \in Y = \{-1, +1\}$; (ii) a weak learner that outputs hypotheses $h : X \to \{-1, +1\}$ and (iii) the unequal loss function $L^\beta(y)$ of Definition 3.1.

We assign initial weight $D_1(i) = w^+$ to the $n^+$ positive examples and $D_1(i) = w^-$ to the $n^-$ negative examples, where $w^+ n^+ + w^- n^- = 1$. The values can be set so that $w^+/w^- = \beta$ or they can be adjusted using a validation set. The generalization of AdaBoost to the case of an unequal loss function is given as the AdaUBoost algorithm in Figure 1. We adapt theorem 1 in [7] for this algorithm.

**Theorem 4.1** *Assuming the notation and algorithm of Figure 1, the following bound holds on the training error of H*

$$w^+|i : H(x_i) \neq y_i = 1| + w^-|i : H(x_i) \neq y_i = -1| \leq \prod_{t=1}^{T} Z_t. \tag{3}$$

The choice of $w^+$ and $w^-$ will force uneven probabilities of misclassification on the training set, but to ensure that the weak learners concentrate on misclassified positive examples we define $Z$ (suppressing the subscript) as

$$Z = \sum_i D(i) \exp(-\alpha \beta_i y_i h(x_i)). \tag{4}$$

Thus, to minimize training error we should seek to minimize $Z$ with respect to $\alpha$ (the voting coefficient) on each iteration of boosting. Following [7], we introduce the notation $W_{++}, W_{-+}, W_{+-}$ and $W_{--}$, where for $s_1$ and $s_2 \in \{-1, +1\}$

$$W_{s_1 s_2} = \sum_{i : y_i = s_1, h(x_i) = s_2} D(i) \tag{5}$$

By equating to zero the first derivative of (4) with respect to $\alpha$, $Z'(\alpha)$, and using (5) we have $-\exp(-\alpha/\beta)W_{++}/\beta + \exp(\alpha/\beta)W_{-+}/\beta + \exp(\alpha)W_{+-} - \exp(-\alpha)W_{--} = 0$. Letting $Y = \exp(\alpha)$ we get a polynomial in $Y$:

$$C_1 Y^{1-1/\beta} + C_2 Y^{1+1/\beta} + C_3 Y^2 + C_4 = 0 \qquad (6)$$

where $C_1 = -W_{++}/\beta$, $C_2 = W_{-+}/\beta$, $C_3 = W_{+-}$, and $C_4 = -W_{--}$.

The root of this polynomial can be found numerically. Since $Z''(\alpha) > 0$, $Z'(\alpha)$ can have at most one zero and this gives the unique minimum of $Z(\alpha)$. The solution for $\alpha$ from (6) is used (as $\alpha_t$) when taking the distance of a training example from the standard threshold on each iteration of the AdaUBoost algorithm in Figure 1 as well as when combining the weak learners in $H(x)$.

The ThetaBoost algorithm searches for a positive and a negative support vector (SV) point such that the hyperplane separating them has the largest margin. Once these SV points are found we can then apply the formulas (1) and (2) of Sections 3.1 and 3.2 respectively to compute values for adjusting the threshold. See Figure 2 for the complete algorithm.

Algorithm AdaUBoost$(X, Y, \beta)$

    1. Initialize $D_1(i)$ as described above.

    2. For $t = 1, ..., T$

        • train weak learner using distribution $D_t$;

        • get weak hypothesis $h_t$;

        • choose $\alpha_t \in \mathbb{R}$;

        • update: $D_{t+1}(i) = D_t(i) \exp[-\alpha_t \beta_i y_i h(x_i)]/Z_t$

        • where $\beta_i = 1/\beta$ if $y_i = 1$ and 1 if otherwise, and $Z_t$ is a normalization factor such that $\sum_i D_{t+1}(i) = 1$;

    3. Output the final hypothesis: $H(x) = \text{sgn}\left(\sum_{t=1}^{T} \alpha_t h_t(x)\right)$.

Algorithm ThetaBoost$(X, Y, \beta, \delta_M)$

    1. $H(x) = \text{AdaUBoost}(X, Y, \beta)$;

    2. Remove from the training dataset the false positive and borderline points;

    3. Find the smallest $H(x_+)$ and mark this as the $SV_+$; and remove any negative points with value greater than $H(SV_+)$;

    4. Find the first negative point that is next in ranking to the $SV_+$ and mark this as $SV_-$; and compute the margin as the sum of distances, $d_+$ and $d_-$, of $SV_+$ and $SV_-$ from the standard threshold;

    5. Check for candidate $SV_-$'s that are near to the current one and change the margin by at least $\delta_M$;

    6. Use $SV_+$ and $SV_-$ to compute the theta threshold from Eqn (1) and (2);

    7. Output the final hypothesis: $H(x) = \text{sgn}\left(\sum_{t=1}^{T} \alpha_t h_t(x) - \theta\right)$

Figure 1: The AdaUBoost and Theta-Boost algorithms.

## 5  Experiments

The purpose of the experiments reported in this section is two-fold:

(i) to compare the generalization performance of AdaUBoost against that of standard Adaboost on imbalanced datasets;

(ii) to examine the two formulas for choosing the threshold in ThetaBoost and evaluate their effect on generalization performance.

For the evaluations in (i) and (ii) we use two performance measures: the average $L^\beta$ and the geometric mean of accuracy (g-mean) [4]. The latter is defined as $g = \sqrt{\text{precision} \cdot \text{recall}}$, where

$$\text{precision} = \frac{\# \text{ positives correct}}{\# \text{ positives predicted}}; \quad \text{recall} = \frac{\# \text{ positives correct}}{\# \text{ true positives}}.$$

The g-mean has recently been proposed as a performance measure that, in contrast to accuracy, can capture the "specificity" trade-off between false positives and true positives in imbalanced datasets [4]. It is also independent of the distribution of examples between classes.

For our initial experiments we used the satimage dataset from the UCI repository [5] and used a uniform $D_1$. The dataset is about classifying neigborhoods of pixels in a satelite image. It has 36 continuous attributes and 6 classes. We picked class 4 as the goal class since it is the less prevalent one (9.73% of the dataset). The dataset comes in a training (4435 examples) and a test (2000 examples) set.

Table 1 shows the performance on the test set of AdaUBoost, AdaBoost and C4.5 for different values of the beta parameter. It should be pointed out that the latter two algorithms minimize the total error assuming an equal loss function ($\beta = 1$). In the case of equal loss AdaUBoost simply reduces to AdaBoost. As observed from the table the higher the loss parameter the bigger the improvement of AdaUBoost over the other two algorithms. This is particularly apparent in the values of g-mean.

| | AdaUBoost | | AdaBoost | | C4.5 | |
|---|---|---|---|---|---|---|
| $\beta$ values | avgLoss | g-mean | avgLoss | g-mean | avgLoss | g-mean |
| 1 | 0.0545 | 0.773 | 0.0545 | 0.773 | 0.0885 | 0.724 |
| 2 | 0.0895 | 0.865 | 0.0831 | 0.773 | 0.136 | 0.724 |
| 4 | 0.13 | 0.889 | 0.1662 | 0.773 | 0.231 | 0.724 |
| 8 | 0.1785 | 0.898 | 0.3324 | 0.773 | 0.421 | 0.724 |
| 16 | 0.267 | 0.89 | 0.664 | 0.773 | 0.801 | 0.724 |

Table 1: Generalization performance in the SatImage dataset.

Figure 2 shows the generalization performance of ThetaBoost in terms of average loss ($\beta = 2$) for different values of the threshold $\theta$. The latter ranges from the largest margin of negative examples that corresponds to $SV_-$ to the smallest margin of positive examples that corresponds to $SV_+$. This range includes the values of $b$ and $\eta$ given by formulas (1) and (2). In this experiment $\delta_M$ was set to 0.2. As depicted in the figure, the margin defined by $b$ achieves better generalization performance than the margin defined by $\eta$. In particular, $b$ is closer to the value of $\theta$ that gives the minimum loss on this test set. In addition, ThetaBoost with $b$ performs better than AdaUBoost on this test set. We should emphasise, however, that the differences are not significant and that more extensive experiments are required before the two approaches can be ranked reliably.

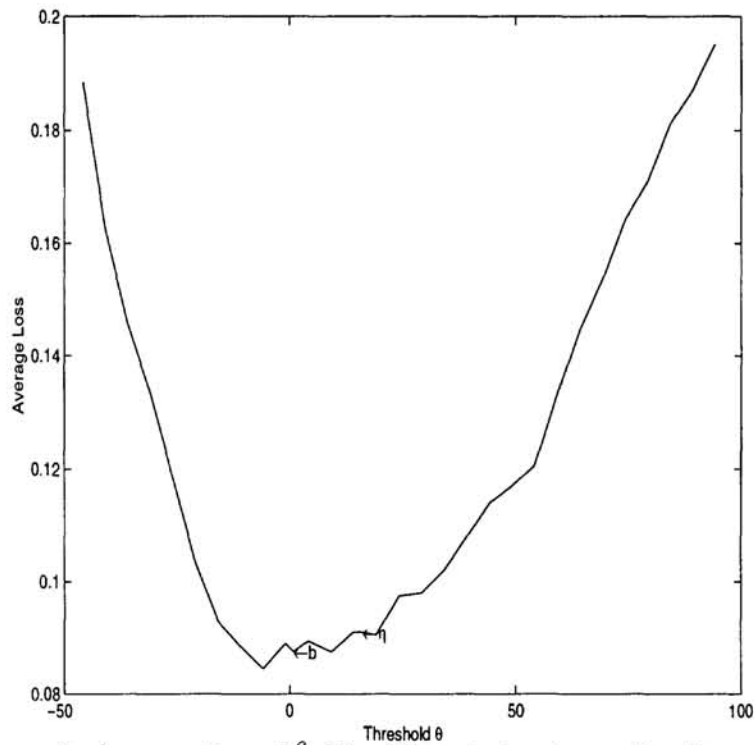

Figure 2: Average Loss $L^\beta$ $(\beta = 2)$ on test set as a function of $\theta$

## 6   Discussion

In the above we built a theoretical framework for optimally setting the margin given an unequal loss function. By applying this framework to boosting we developed AdaUBoost and ThetaBoost that generalize Adaboost, a well known boosting algorithm, for taking into account unequal loss functions and adjusting the margin in imbalanced datasets. Initial experiments have shown that both these factors improve the generalization performance of the boosted classifier.

## References

[1] Corinna Cortes and Vladimir Vapnik, *Machine Learning*, **20**, 273–297, 1995.

[2] Yoav Freund and Robert Schapire, pages 148–156 in *Proceedings of the International Conference on Machine Learning, ICML'96*, 1996.

[3] Michael J. Kearns and Robert E. Schapire, pages 382–391 in *Proceedings of the 31st Symposium on the Foundations of Computer Science, FOCS'90*, 1990.

[4] Kubat, M., Holte, R. and Matwin, S., Machine Learning, **30**, 195-215, 1998.

[5] Merz, C.J. and Murphy, P.M. (1997). UCI repository of machine learning databases. http://www.ics.uci.edu/ mlearn/MLRepository.html.

[6] R. Schapire, Y. Freund, P. Bartlett, W. Sun Lee, pages 322–330 in *Proceedings of International Conference on Machine Learning, ICML'97*, 1997.

[7] Robert Schapire and Yoram Singer, in *Proceedings of the Eleventh Annual Conference on Computational Learning Theory, COLT'98*, 1998.

[8] John Shawe-Taylor, Algorithmica, **22**, 157–172, 1998.

[9] John Shawe-Taylor, Peter Bartlett, Robert Williamson and Martin Anthony, IEEE Trans. Inf. Theory, **44** (5) 1926–1940, 1998.
